# Spatial Representations in the Parietal Cortex May Use Basis Functions

**Alexandre Pouget**
alex@salk.edu

**Terrence J. Sejnowski**
terry@salk.edu

Howard Hughes Medical Institute
The Salk Institute
La Jolla, CA 92037
and
Department of Biology
University of California, San Diego

## Abstract

The parietal cortex is thought to represent the egocentric positions of objects in particular coordinate systems. We propose an alternative approach to spatial perception of objects in the parietal cortex from the perspective of sensorimotor transformations. The responses of single parietal neurons can be modeled as a gaussian function of retinal position multiplied by a sigmoid function of eye position, which form a set of basis functions. We show here how these basis functions can be used to generate receptive fields in either retinotopic or head-centered coordinates by simple linear transformations. This raises the possibility that the parietal cortex does not attempt to compute the positions of objects in a particular frame of reference but instead computes a general purpose representation of the retinal location and eye position from which any transformation can be synthesized by direct projection. This representation predicts that hemineglect, a neurological syndrome produced by parietal lesions, should not be confined to egocentric coordinates, but should be observed in multiple frames of reference in single patients, a prediction supported by several experiments.

## 1 Introduction

The temporo-parietal junction in the human cortex and its equivalent in monkeys, the inferior parietal lobule, are thought to play a critical role in spatial perception. Lesions in these regions typically result in a neurological syndrome, called hemineglect, characterized by a lack of motor exploration toward the hemispace contralateral to the site of the lesion. As demonstrated by Zipser and Andersen [11], the responses of single cells in the monkey parietal cortex are also consistent with this presumed role in spatial perception.

In the general case, recovering the egocentric position of an object from its multiple sensory inputs is difficult because of the multiple reference frames that must be integrated. In this paper, we consider a simpler situation in which there is only visual input and all body parts are fixed but the eyes, a condition which has been extensively used for neurophysiological studies in monkeys. In this situation, the head-centered position of an object, $\vec{A}$, can be readily recovered from the retinal location, $\vec{R}$, and current eye position, $\vec{E}$, by vector addition:

$$\vec{A} = \vec{R} + \vec{E} \qquad (1)$$

If the parietal cortex contains a representation of the egocentric position of objects, then one would expect to find a representation of the vectors, $\vec{A}$, associated with these objects. There is an extensive literature on how to encode a vector with a population of neurons, and we first present two schemes that have been or are used as working hypothesis to study the parietal cortex. The first scheme involves what is typically called a computational map, whereas the second uses a vectorial representation [9].

This paper shows that none of these encoding schemes accurately accounts for all the response properties of single cells in the parietal cortex. Instead, we propose an alternative hypothesis which does not aim at representing $\vec{A}$ *per se*; instead, the inputs $\vec{R}$ and $\vec{E}$ are represented in a particular basis function representation. We show that this scheme is consistent with the way parietal neurons respond to the retinal position of objects and eye position, and we give computational arguments for why this might be an efficient strategy for the cortex.

## 2 Maps and Vectorial Representations

One way to encode a two-dimensional vector is to use a lookup table for this vector which, in the case of a two-dimensional vector, would take the form of a two-dimensional neuronal map. The parietal cortex may represent the egocentric location of object, $\vec{A}$, in a similar fashion. This predicts that the visual receptive field of parietal neurons have a fixed position with respect to the head (figure 1B). The work of Andersen *et al.* (1985) have clearly shown that this is not the case. As illustrated in figure 2A, parietal neurons have retinotopic receptive fields.

In a vectorial representation, a vector is encoded by N units, each of them coding for the projection of the vector along its preferred direction. This entails that the activity, $h$, of a neuron is given by:

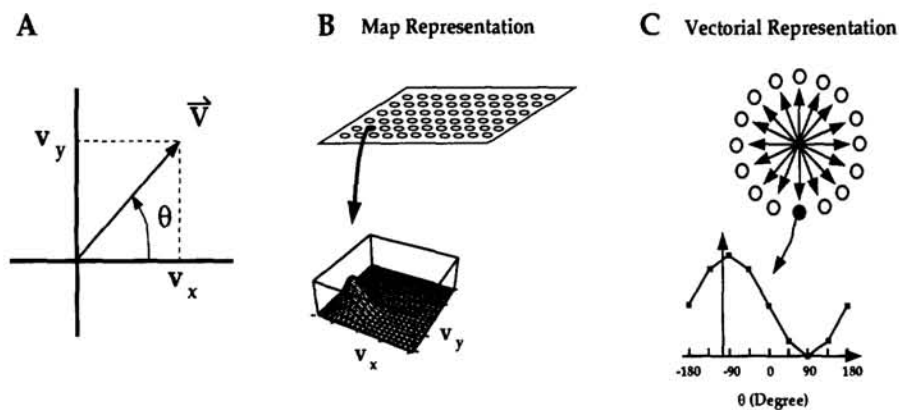

Figure 1: Two neural representations of a vector. A) A vector $\vec{V}$ in cartesian and polar coordinates. B) In a map representation, units have a narrow gaussian tuning to the horizontal and vertical components of $\vec{V}$. Moreover, the position of the peak response is directly related to the position of the units on the map. C) In a vectorial representation, each unit encodes the projection of $\vec{V}$ along its preferred direction (central arrows). This results in a cosine tuning to the vector angle, $\theta$ .

$$h = \vec{W}_a^T \vec{A} = \|\vec{W}_a\| \, \|\vec{A}\| \cos \theta \qquad (2)$$

$\vec{W}_a$ is usually called the preferred direction of the cells because the activity is maximum whenever $\theta = 0$; that is, when $\vec{A}$ points in the same direction as $\vec{W}_a$. Such neurons have a cosine tuning to the direction of the egocentric location of objects, as shown also in figure 1C.

Cosine tuning curves have been reported in the motor cortex by Georgopoulos *et al.* (1982), suggesting that the motor cortex uses a vectorial code for the direction of hand movement in extrapersonal space. The same scheme has been also used by Goodman and Andersen (1990), and Touretzski *et al.* (1993) to model the encoding of egocentric position of objects in the parietal cortex. Touretzski *et al.* (1993) called their representation a *sinusoidal array* instead of a *vectorial representation*.

Using Eq. 1, we can rewrite Eq. 2:

$$h = \quad \vec{W}_a^T(\vec{R} + \vec{E}) = \vec{W}_a^T \vec{R} + \vec{W}_a^T \vec{E} \qquad (3)$$

This second equation is linear in $\vec{R}$ and $\vec{E}$ and uses the same vectors, $\vec{W}_a$, in both dot products. This leads to three important predictions:

1) The visual receptive fields of parietal neurons should be planar.

2) The eye position receptive fields of parietal neurons should also be planar; that is, for a given retinal positions, the response of parietal neuron should be a linear function of eye position.

**A**                                                    **B**

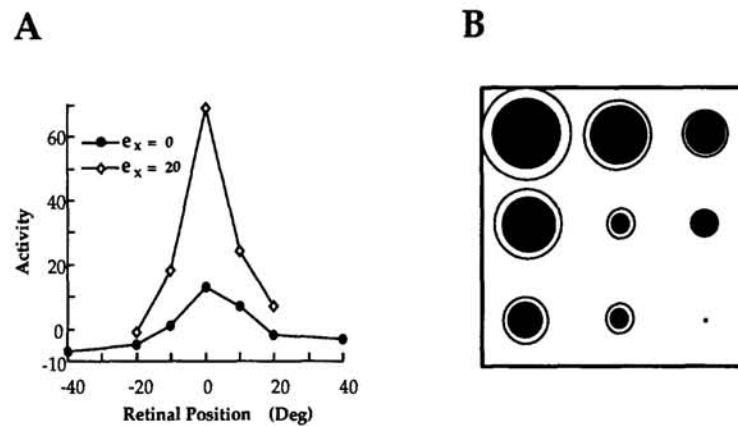

Figure 2: Typical response of a neuron in the parietal cortex of a monkey. A) Visual receptive field has a fixed position on the retina, but the gain of the response is modulated by eye position ($e_x$). (Adpated from Andersen *et al.*, 1985) B) Example of an eye position receptive field, also called gain field, for a parietal cell. The nine circles indicate the amplitude of the response to an identical retinal stimulation for nine different eye positions. Outer circles show the total activity, whereas black circles correspond to the total response minus spontaneous activity prior to visual stimulation. (Adpated from Zipser *et al.*, 1988)

3) The preferred direction for retinal location and eye position should be identical. For example, if the receptive field is on the right side of the visual field, the gain field should also increase with eye positon to the right side.

The visual receptive fields and the eye position gain fields of single parietal neurons have been extensively studied by Andersen *et al.* [2]. In most cases, the visual receptive fields were bell-shaped with one or several peaks and an average radius of 22 degrees of visual angle [1], a result that is clearly not consistent with the first prediction above. We show in figure 2A an idealized visual receptive field of a parietal neuron. The effect of eye position on the visual receptive field is also illustrated. The eye position clearly modulates the gain of the visual response.

The prediction regarding the receptive field for eye position has been borne out by statistical analysis. The gain fields of 80% of the cells had a planar component [1, 11]. One such gain field is shown in figure 2B.

There is not enough data available to determine whether or not the third prediction is valid. However, indirect evidence suggests that if such a correlation exists between preferred direction for retinal location and for eye position, it is probably not strong. Cells with *opposite* preferred directions [2, 3] have been observed. Furthermore, although each hemisphere represents all possible preferred eye position directions, there is a clear tendency to overrepresent the contralateral retinal hemifield [1].

In conclusion, the experimental data are not fully consistent with the predictions of the vectorial code. The visual receptive fields, in particular, are strongly nonlinear. If these nonlinearities are computationally neutral, that is, they are averaged out in subsequent stages of processing in the cortex, then the vectorial code could capture

the essence of what the parietal cortex computes and, as such, would provide a valid approximation of the neurophysiological data. We argue in the next section that the nonlinearities cannot be disregarded and we present a representational scheme in which they have a central computational function.

# 3 Basis Function Representation

## 3.1 Sensorimotor Coordination and Nonlinear Function Approximation

The function which specified the pattern of muscle activities required to move a limb, or the body, to a specific spatial location is a highly nonlinear function of the sensory inputs. The cortex is not believed to specify patterns of muscle activation, but the intermediate transformations which are handled by the cortex are often themselves nonlinear. Even if the transformations are actually linear, the nature of cortical representations often makes the problem a nonlinear mapping. For example, there exists in the putamen and premotor cortex cells with gaussian head-centered visual receptive fields [7] which means that these cells compute gaussians of $\vec{A}$ or, equivalently, gaussians of $\vec{R} + \vec{E}$, which is nonlinear in $\vec{R}$ and $\vec{E}$. There are many other examples of sensory remappings involving similar computations. If the parietal cortex is to have a role in these remappings, the cells should respond to the sensory inputs in a way that can be used to approximate the nonlinear responses observed elsewhere.

One possibility would be for parietal neurons to represent input signals such as eye position and retinal location with basis functions. A basis function decomposition is a well-known method for approximating nonlinear functions which is, in addition, biologically plausible [8]. In such a representation, neurons do not encode the head-centered locations of objects, $\vec{A}$; instead, they compute functions of the input variables, such as $\vec{R}$ and $\vec{E}$, which can be used subsequently to approximate any functions of these variables.

## 3.2 Predictions of the Basis Function Representation

Not all functions are basis functions. Linear functions do not qualify, nor do sums of functions which, individually, would be basis functions, such as gaussian functions of retinal location plus a sigmoidal functions of eye position. If the parietal cortex uses a basis function representation, two conditions have to be met:

1) The visual and the eye position receptive fields should be smooth nonlinear function of $\vec{R}$ and $\vec{E}$.

2) The selectivities to $\vec{R}$ and $\vec{E}$ should interact nonlinearly

The visual receptive fields of parietal neurons are typically smooth and nonlinear. Gaussian or sum of gaussians appear to provide good models of their response profiles [2]. The eye position receptive field on the other hand, which is represented by the gain field, appears to be approximately linear. We believe, however, that the published data only demonstrate that the eye position receptive field is monotonic,

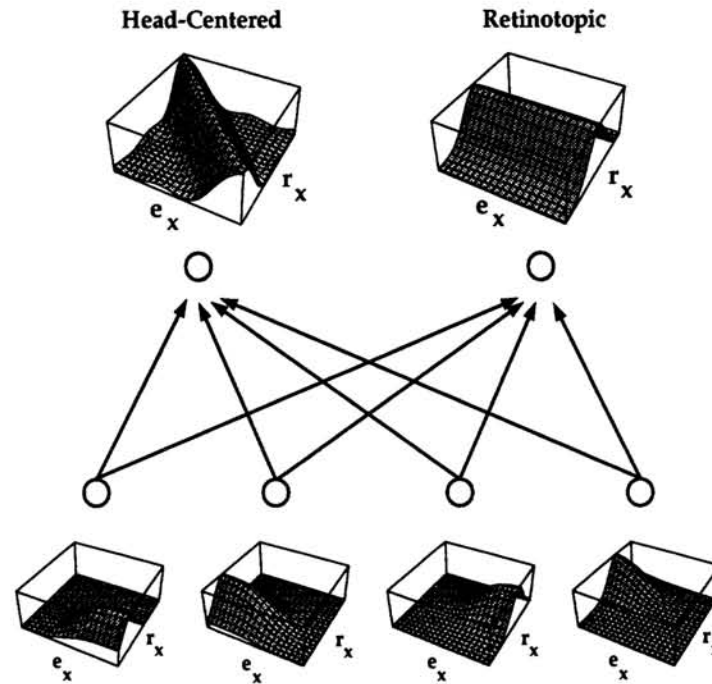

Figure 3: Approximation of a gaussian head-centered (top-left) and a retinotopic (top-right) receptive field, by a linear combination of basis function neurons. The bottom 3-D plots show the response to all possible horizontal retinal position, $r_x$, and horizontal eye positions, $e_x$, of four typical basis function units. These units are meant to model actual parietal neurons

but not necessarily linear. In published experiments, eye position receptive fields (gain fields) were sampled at only nine points, which makes it difficult to distinguish between a plane and other functions such as a sigmoidal function or a piecewise linear function. The hallmark of a nonlinearity would be evidence for saturation of activity within working range of eye position. Several published gain fields show such saturations [3, 11], but a rigorous statistical analysis would be desirable.

Andersen *et al.* (1985) have have shown that the responses of parietal neurons are best modeled by a multiplication between the retinal and eye position selectivities which is consistent with the requirements for basis functions.

Therefore, the experimental data are consistent with our hypothesis that the parietal cortex uses a basis function representation. The response of most gain-modulated neurons in the parietal cortex could be modeled by multiplying a gaussian tuning to retinal position by a sigmoid of eye position, a function which qualifies as a basis function.

## 3.3  Simulations

We simulated the response of 121 parietal gain-modulated neurons modeled by multiplying a gaussian of retinal position, $r_x$, with a sigmoid of eye position, $e_x$:

$$h_i = \frac{e^{-\frac{(r_x - r_{xi})^2}{2\sigma^2}}}{1 + e^{-\frac{e_x - e_{xi}}{t}}} \tag{4}$$

where the centers of the gaussians for retinal loction $r_{xi}$ and the positions of the sigmoids for eye postions $e_{xi}$ were uniformly distributred. The widths of the gaussian $\sigma$ and the sigmoid $t$ were fixed. Four of these functions are shown at the bottom of figure 3.

We used these basis functions as a hidden layer to approximate two kinds of output functions: a gaussian head-centered receptive field and a gaussian retinotopic receptive field. Neurons with these response properties are found downstream of the parietal cortex in the premotor cortex [7] and superior colliculus, two structures believed to be involved in the control of, respectively, arm and eye movements.

The weights for a particular output were obtained by using the delta rule. Weights were adjusted until the mean error was below 5% of the maximum output value. Figure 3 shows our best approximations for both the head-centered and retinotopic receptive fields. This demonstrates that the same pool of neurons can be used to approximate several diffferent nonlinear functions.

## 4 Discussion

Neurophysiological data support our hypothesis that the parietal cortex represents its inputs, such as the retinal location of objects and eye position, in a format suitable to non-linear function approximation, an operation central to sensorimotor coordination. Neurons have gaussian visual receptive fields modulated by monotonic function of eye position leading to response function that can be modeled by product of gaussian and sigmoids. Since the product of gaussian and sigmoids forms basis functions, this representation is good for approximating nonlinear functions of the input variables.

Previous attempts to characterize spatial representations have emphasized linear encoding schemes in which the location of objects is represented in egocentric coordinates. These codes cannot be used for nonlinear function approximation and, as such, may not be adequate for sensorimotor coordination [6, 10]. On the other hand, such representations are computationally interesting for certain operations, like addition or rotation. Some part of the brain more specialized in navigation like the hippocampus might be using such a scheme [10].

In figure 3, a head-centered or a retinotopic receptive field can be computed from the same pool of neurons. It would be arbitrary to say that these neurons encode the positions of objects in egocentric coordinates. Instead, these units encode a position in several frames of reference simultaneously. If the parietal cortex uses this basis function representation, we predict that hemineglect, the neurological syndrome which results from lesions in the parietal cortex, should not be confined to any particular frame of reference. This is precisely the conclusion that has emerged from recent studies of parietal patients [4]. Whether the behavior of parietal patients can be fully explained by lesions of a basis function representation remains to be investigated.

**Acknowledgments**

We thank Richard Andersen for helpful conversations and with access to unpublished data.

# References

[1] R.A. Andersen, C. Asanuma, G. Essick, and R.M. Siegel. Corticocortical connections of anatomically and physiologically defined subdivisions within the inferior parietal lobule. *Journal of Comparative Neurology*, 296(1):65–113, 1990.

[2] R.A. Andersen, G.K. Essick, and R.M. Siegel. Encoding of spatial location by posterior parietal neurons. *Science*, 230:456–458, 1985.

[3] R.A. Andersen and D. Zipser. The role of the posterior parietal cortex in coordinate transformations for visual-motor integration. *Canadian Journal of Physiology and Pharmacology*, 66:488–501, 1988.

[4] M. Behrmann and M. Moscovitch. Object-centered neglect in patient with unilateral neglect: effect of left-right coordinates of objects. *Journal of Cognitive Neuroscience*, 6:1–16, 1994.

[5] A.P. Georgopoulos, J.F. Kalaska, R. Caminiti, and J.T. Massey. On the relations between the direction of two-dimensional arm movements and cell discharge in primate motor cortex. *Journal of Neuroscience*, 2(11):1527–1537, 1982.

[6] S.J. Goodman and R.A. Andersen. Algorithm programmed by a neural model for coordinate transformation. In *International Joint Conference on Neural Networks*, San Diego, 1990.

[7] M.S. Graziano, G.S. Yap, and C.G. Gross. Coding of visual space by premotor neurons. *Science*, 266:1054–1057, 1994.

[8] T. Poggio. A theory of how the brain might work. *Cold Spring Harbor Symposium on Quantitative Biology*, 55:899–910, 1990.

[9] J.F. Soechting and M. Flanders. Moving in three-dimensional space: frames of reference, vectors and coordinate systems. *Annual Review in Neuroscience*, 15:167–91, 1992.

[10] D.S. Touretzky, A.D. Redish, and H.S. Wan. Neural representation of space using sinusoidal arrays. *Neural Computation*, 5:869–884, 1993.

[11] D. Zipser and R.A. Andersen. A back-propagation programmed network that stimulates reponse properties of a subset of posterior parietal neurons. *Nature*, 331:679–684, 1988.